# Discontinuous Recall Transitions Induced By Competition Between Short- and Long-Range Interactions in Recurrent Networks

**N.S. Skantzos, C.F. Beckmann and A.C.C. Coolen**
Dept of Mathematics, King's College London, Strand, London WC2R 2LS, UK
E-mail: skantzos@mth.kcl.ac.uk   tcoolen@mth.kcl.ac.uk

## Abstract

We present exact analytical equilibrium solutions for a class of recurrent neural network models, with both sequential and parallel neuronal dynamics, in which there is a tunable competition between nearest-neighbour and long-range synaptic interactions. This competition is found to induce novel coexistence phenomena as well as discontinuous transitions between pattern recall states, 2-cycles and non-recall states.

## 1   INTRODUCTION

Analytically solvable models of large recurrent neural networks are bound to be simplified representations of biological reality. In early analytical studies such as [1, 2] neurons were, for instance, only allowed to interact with a strength which was independent of their spatial distance (these are the so-called mean field models). At present both the statics of infinitely large mean-field models of recurrent networks, as well as their dynamics away from saturation are well understood, and have obtained the status of textbook or review paper material [3, 4]. The focus in theoretical research of recurrent networks has consequently turned to new areas such as solving the dynamics of large networks close to saturation [5], the analysis of finite size phenomenology [6], solving biologically more realistic (e.g. spike-based) models [7] or analysing systems with spatial structure. In this paper we analyse models of recurrent networks with spatial structure, in which there are two types of synapses: long-range ones (operating between any pair of neurons), and short-range ones (operating between nearest neighbours only). In contrast to early papers on spatially structured networks [8], one here finds that, due to the nearest neighbour interactions, exact solutions based on simple mean-field approaches are ruled out. Instead, the present models can be solved exactly by a combination of mean-field techniques and the so-called transfer matrix method (see [9]). In parameter regimes where the two synapse types compete (where one has long-range excitation with short-range inhibition, or long-range Hebbian synapses with short-range anti-Hebbian synapses) we find interesting and potentially useful novel phenomena, such as coexistence of states and discontinuous transitions between them.

## 2   MODEL DEFINITIONS

We study models with $N$ binary neuron variables $\sigma_i = \pm 1$, which evolve in time stochastically on the basis of post-synaptic potentials $h_i(\vec{\sigma})$, following

$$\text{Prob}[\sigma_i(t+1) = \pm 1] = \frac{1}{2}\left[1 \pm \tanh[\beta h_i(\vec{\sigma}(t))]\right] \qquad h_i(\vec{\sigma}) = \sum_{j \neq i} J_{ij}\sigma_j + \theta_i \quad (1)$$

The variables $J_{ij}$ and $\theta_i$ represent synaptic interactions and firing thresholds, respectively. The (non-negative) parameter $\beta$ controls the amount of noise, with $\beta = 0$ and $\beta = \infty$ corresponding to purely random and purely deterministic response, respectively. If the synaptic matrix is symmetric, both a random sequential execution and a fully parallel execution of the stochastic dynamics (1) will evolve to a unique equilibrium state. The corresponding microscopic state probabilities can then both formally be written in the Boltzmann form $p_\infty(\vec{\sigma}) \sim \exp[-\beta H(\vec{\sigma})]$, with [10]

$$H_{\text{seq}}(\vec{\sigma}) = -\sum_{i<j} \sigma_i J_{ij}\sigma_j - \sum_i \theta_i\sigma_i \qquad H_{\text{par}}(\vec{\sigma}) = -\frac{1}{\beta}\sum_i \log\cosh[\beta h_i(\vec{\sigma})] - \sum_i \theta_i\sigma_i$$
$$(2)$$

For large systems the macroscopic observables of interest can be obtained by differentiation of the free energy per neuron $f = -\lim_{N\to\infty}(\beta N)^{-1}\log\sum_{\vec{\sigma}}\exp[-\beta H(\vec{\sigma})]$, which acts as a generating function. For the synaptic interactions $J_{ij}$ and the thresholds $\theta_i$ we now make the following choice:

$$\text{model I}: \qquad J_{ij} = \frac{J_\ell}{N}\xi_i\xi_j + J_s(\delta_{i,j+1} + \delta_{i,j-1})\,\xi_i\xi_j \qquad \theta_i = \theta\xi_i \quad (3)$$

(which corresponds to the result of having stored a binary pattern $\vec{\xi} \in \{-1,1\}^N$ through Hebbian-type learning), with $J_\ell, J_s, \theta \in \Re$ and $i + N \equiv i$. The neurons can be thought of as being arranged on a periodic one-dimensional array, with uniform interactions of strength $J_\ell\xi_i\xi_j/N$, in combination with nearest neighbour interactions of strength $J_s\xi_i\xi_j$. Note that model I behaves in exactly the same way as the following

$$\text{model II}: \qquad J_{ij} = \frac{J_\ell}{N} + J_s(\delta_{i,j+1} + \delta_{i,j-1}) \qquad \theta_i = \theta \quad (4)$$

since a simple transformation $\sigma_i \to \sigma_i\xi_i$ maps one model into the other. Taking derivatives of $f$ with respect to the parameters $\theta$ and $J_s$ for model II produces our order parameters, expressed as equilibrium expectation values. For sequential dynamics we have

$$m = -\frac{\partial f}{\partial\theta} = \lim_{N\to\infty}\frac{1}{N}\sum_i \langle\sigma_i\rangle \qquad a = -\frac{\partial f}{\partial J_s} = \lim_{N\to\infty}\frac{1}{N}\sum_i \langle\sigma_{i+1}\sigma_i\rangle \quad (5)$$

For parallel dynamics the corresponding expressions turn out to be

$$m = -\frac{1}{2}\frac{\partial f}{\partial\theta} = \lim_{N\to\infty}\frac{1}{N}\sum_i \langle\sigma_i\rangle \qquad a = -\frac{1}{2}\frac{\partial f}{\partial J_s} = \lim_{N\to\infty}\frac{1}{N}\sum_i \langle\sigma_{i+1}\tanh[\beta h_i(\vec{\sigma})]\rangle$$
$$(6)$$

We have simplified (6) with the identities $\langle\sigma_{i+1}\tanh[\beta h_i(\vec{\sigma})]\rangle = \langle\sigma_{i-1}\tanh[\beta h_i(\vec{\sigma})]\rangle$ and $\langle\tanh[\beta h_i(\vec{\sigma})]\rangle = \langle\sigma_i\rangle$, which follow from (1) and from invariance under the transformation $i \to N+1-i$ (for all $i$). For sequential dynamics $a$ describes the average equilibrium state covariances of neighbouring neurons. For parallel dynamics $a$ gives the average equilibrium state covariances of neurons *at a given time $t$*, and their neighbours *at time $t+1$* (the difference between the two meanings of $a$ will be important in the presence of 2-cycles). In model II $m$ is the average activity in equilibrium, whereas for model I one finds

$$m = \lim_{N\to\infty}\frac{1}{N}\sum_i \xi_i\langle\sigma_i\rangle$$

This is the familiar overlap order parameter of associative memory models [1, 2], which measures the quality of pattern recall in equilibrium. The observable $a$ transforms similarly.

# 3  SOLUTION VIA TRANSFER MATRICES

From this stage onwards our analysis will refer to model II, i.e eqn (4); the results can immediately be translated into the language of model I (3) via the transformation $\sigma_i \to \sigma_i \xi_i$. In calculating $f$ it is advantageous to separate terms induced by the long-range synapses from those induced by the short-range ones, via insertion of $1 = \int dm\, \delta[m - \frac{1}{N}\sum_i \sigma_i]$. Upon using the integral representation of the $\delta$-function, we then arrive at

$$f = -\lim_{N\to\infty} \frac{1}{\beta N} \log \int dm\, d\hat{m}\, e^{-\beta N \phi(m,\hat{m})}$$

with

$$\phi_{\text{seq}}(m,\hat{m}) = -im\hat{m} - m\theta - \frac{1}{2}J_\ell m^2 - \frac{1}{\beta N} \log R_{\text{seq}}(\hat{m}) \qquad (7)$$

$$\phi_{\text{par}}(m,\hat{m}) = -im\hat{m} - m\theta - \frac{1}{\beta N} \log R_{\text{par}}(m,\hat{m}) \qquad (8)$$

The quantities $R$ contain all complexities due to the short-range interactions in the model (they describe a periodic one-dimensional system with neighbour interactions only):

$$R_{\text{seq}}(\hat{m}) = \sum_{\vec{\sigma}\in\{-1,1\}^N} e^{-i\beta\hat{m}\sum_i \sigma_i}\, e^{\beta J_s \sum_i \sigma_i \sigma_{i+1}}$$

$$R_{\text{par}}(m,\hat{m}) = \sum_{\vec{\sigma}\in\{-1,1\}^N} e^{-i\beta\hat{m}\sum_i \sigma_i}\, e^{\sum_i \log\cosh\beta[J_\ell m+\theta+J_s(\sigma_{i+1}+\sigma_{i-1})]}$$

They can be calculated using the transfer-matrix technique [9], which exploits an interpretation of the summations over the $N$ neuron states $\sigma_i$ as matrix multiplications, giving

$$R_{\text{seq}}(\hat{m}) = \text{Tr}\,[T_{\text{seq}}^N] \qquad T_{\text{seq}} = \begin{pmatrix} e^{\beta J_s - i\beta\hat{m}} & e^{-\beta J_s} \\ e^{-\beta J_s} & e^{\beta J_s + i\beta\hat{m}} \end{pmatrix}$$

$$R_{\text{par}}(m,\hat{m}) = \text{Tr}\,[T_{\text{par}}^N] \qquad T_{\text{par}} = \begin{pmatrix} \cosh[\beta w_+]e^{-i\beta\hat{m}} & \cosh[\beta w_0] \\ \cosh[\beta w_0] & \cosh[\beta w_-]e^{i\beta\hat{m}} \end{pmatrix}$$

where $w_0 = J_\ell m + \theta$ and $w_\pm = w_0 \pm 2J_s$. The identity $\text{Tr}\,[T^N] = \lambda_+^N + \lambda_-^N$, in which $\lambda_\pm$ are the eigenvalues of the $2 \times 2$ matrix $T$, enables us to take the limit $N \to \infty$ in our equations. The integral over $(m,\hat{m})$ is for $N \to \infty$ evaluated by gradient descent, and is dominated by the saddle points of the exponent $\phi$. We thus arrive at the transparent result

$$f = \text{extr}\,\phi(m,\hat{m}) \qquad \begin{cases} \phi_{\text{seq}}(m,\hat{m}) = -im\hat{m} - m\theta - \frac{1}{2}J_\ell m^2 - \frac{1}{\beta}\log\lambda_+^{\text{seq}} \\ \phi_{\text{par}}(m,\hat{m}) = -im\hat{m} - m\theta - \frac{1}{\beta}\log\lambda_+^{\text{par}} \end{cases} \qquad (9)$$

where $\lambda_+^{\text{seq}}$ and $\lambda_+^{\text{par}}$ are the largest eigenvalues of $T_{\text{seq}}$ and $T_{\text{par}}$. For simplicity, we will restrict ourselves to the case where $\theta = 0$; generalisation of what follows to the case of arbitrary $\theta$, by using the full form of (9), is not significantly more difficult. The expressions defining the value(s) of the order parameter $m$ can now be obtained from the saddle point equations $\partial_m \phi(m,\hat{m}) = \partial_{\hat{m}}\phi(m,\hat{m}) = 0$. Straightforward differentiation shows that

$$\text{sequential}: \qquad \hat{m} = imJ_\ell, \qquad m = G(m; J_\ell, J_s)$$

$$\text{parallel}: \qquad \hat{m} = imJ_\ell, \qquad m = G(m; J_\ell, J_s) \qquad \text{for } J_\ell \geq 0 \qquad (10)$$

$$\hat{m} = -imJ_\ell, \qquad m = G(m; -J_\ell, -J_s) \qquad \text{for } J_\ell < 0$$

with

$$G(m; J_\ell, J_s) = \frac{\sinh[\beta J_\ell m]}{\sqrt{\sinh^2[\beta J_\ell m] + e^{-4\beta J_s}}} \qquad (11)$$

Note that equations (10,11) allow us to derive the physical properties of the parallel dynamics model from those of the sequential dynamics model via simple transformations.

## 4 PHASE TRANSITIONS

Our main order parameter $m$ is to be determined by solving an equation of the form $m = G(m)$, in which $G(m) = G(m; J_\ell, J_s)$ for both sequential and parallel dynamics with $J_\ell \geq 0$, whereas $G(m) = G(m; -J_\ell, -J_s)$ for parallel dynamics with $J_\ell < 0$. Note that, due to $G(0; J_\ell, J_s) = 0$, the trivial solution $m = 0$ always exists. In order to obtain a phase diagram we have to perform a bifurcation analysis of the equations (10,11), and determine the combinations of parameter values for which specific non-zero solutions are created or annihilated (the transition lines). Bifurcations of non-zero solutions occur when

$$m = G(m) \qquad \text{and} \qquad 1 = G'(m) \tag{12}$$

The first equation in (12) states that $m$ must be a solution of the saddle-point problem, the second one states that this solution is in the process of being created/annihilated. Nonzero solutions of $m = G(m)$ can come into existence in two qualitatively different ways: as continuous bifurcations away from the trivial solution $m = 0$, and as discontinuous bifurcations away from the trivial solution. These two types will have to be treated differently.

### 4.1  Continuous Transitions

An analytical expression for the lines in the $(\beta J_s, \beta J_\ell)$ plane where continuous transitions occur between recall states (where $m \neq 0$) and non-recall states (where $m = 0$) is obtained by solving the coupled equations (12) for $m = 0$. This gives:

$$cont.\ trans. : \qquad \begin{array}{ll} \text{sequential}: & \beta J_\ell = e^{-2\beta J_s} \\ \text{parallel}: & \beta J_\ell = e^{-2\beta J_s} \quad \text{and} \quad \beta J_\ell = -e^{2\beta J_s} \end{array} \tag{13}$$

If along the transition lines (13) we inspect the behaviour of $G(m)$ close to $m = 0$ we can anticipate the possible existence of discontinuous ones, using the properties of $G(m)$ for $m \to \pm\infty$, in combination with $G(-m) = -G(m)$. Precisely *at* the lines (13) we have $G(m) = m + \frac{1}{6}G'''(0).m^3 + \mathcal{O}(m^5)$. Since $\lim_{m\to\infty} G(m) = 1$ one knows that for $G'''(0) > 0$ the function $G(m)$ will have to cross the diagonal $G(m) = m$ again at some value $m > 0$ in order to reach the limit $G(\infty) = 1$. This implies, in combination with $G(-m) = -G(m)$, that a discontinous transition must have already taken place earlier, and that away from the lines (13) there will consequently be regions where one finds five solutions of $m = G(m)$ (two positive ones, two negative ones). Along the lines (13) the condition $G'''(0) > 0$, pointing at discontinuous transitions elsewhere, translates into

$$\begin{array}{lll} \text{sequential}: & \beta J_\ell > \sqrt{3} & \text{and} \quad \beta J_s < -\frac{1}{4}\log 3 \\ \text{parallel}: & |\beta J_\ell| > \sqrt{3} & \text{and} \quad |\beta J_s| < -\frac{1}{4}\log 3 \end{array} \tag{14}$$

### 4.2  Discontinuous Transitions

In the present models it turns out that one can also find an analytical expression for the discontinuous transition lines in the $(\beta J_s, \beta J_\ell)$ plane, in the form of a parametrisation. For sequential dynamics one finds a single line, parametrised by $x = \beta J_\ell m \in [0, \infty)$:

$$discont.\ trans. : \qquad \beta J_\ell(x) = \sqrt{\frac{x^3}{x - \tanh(x)}}, \qquad \beta J_s(x) = -\frac{1}{4}\log\left[\frac{\tanh(x)\sinh^2(x)}{x - \tanh(x)}\right] \tag{15}$$

Since this parametrisation (15) obeys $\beta J_s(0) = -\frac{1}{4}\log 3$ and $\beta J_\ell(0) = \sqrt{3}$, the discontinuous transition indeed starts precisely at the point predicted by the convexity of $G(m)$ at $m = 0$, see (14). For sequential dynamics the line (15) gives all non-zero solutions of the coupled equations (12). For parallel dynamics one finds, in addition to (15), a second 'mirror image' transition line, generated by the transformation $\{\beta J_\ell, \beta J_s\} \mapsto \{-\beta J_\ell, -\beta J_s\}$.

# 5 PHASE DIAGRAMS

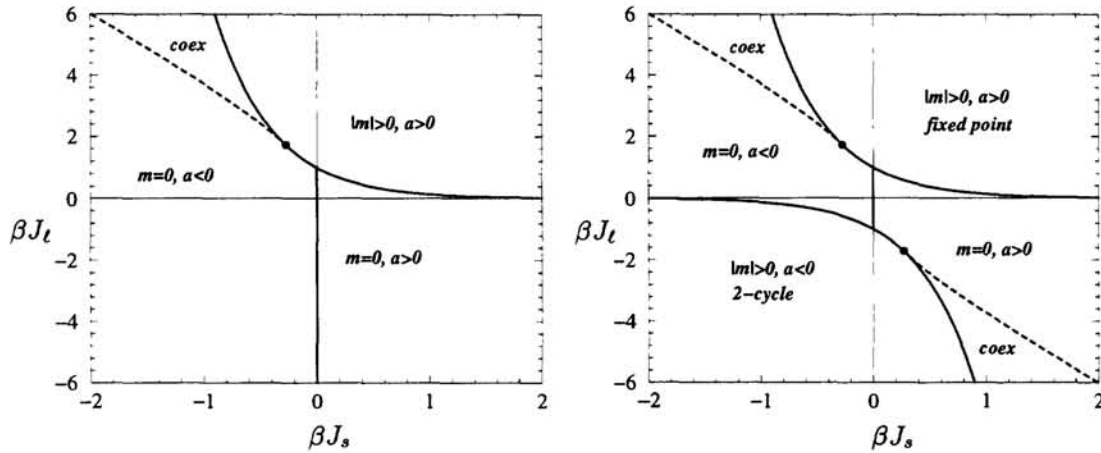

Figure 1: Left: phase diagram for sequential dynamics, involving three regions: ($i$) a region with $m = 0$ only (here $a = \tanh[\beta J_s]$), ($ii$) a region with two $m \neq 0$ fixed-point states (with opposite sign, and with identical $a > 0$), and ($iii$) a region where the $m = 0$ state and the two $m \neq 0$ states coexist. The ($i$) $\rightarrow$ ($ii$) and ($ii$) $\rightarrow$ ($iii$) transitions are continuous (solid lines), whereas the ($i$) $\rightarrow$ ($iii$) transition is discontinuous (dashed line). Right: phase diagram for parallel dynamics, involving the above regions and transitions, as well as a second set of transition lines (in the region $J_\ell < 0$) which are exact reflections in the origin of the first set. Here, however, the $m = 0$ region has $a = \tanh[2\beta J_s]$, the two $m \neq 0$ physical solutions describe 2-cycles rather than fixed-points, and the $J_\ell < 0$ coexistence region describes the coexistence of an $m = 0$ fixed-point and 2-cycles.

Having determined the transition lines in parameter space, we can turn to the phase diagrams. A detailed expose of the various procedures followed to determine the nature of the various phases, which are also dependent on the type of dynamics used, goes beyond the scope of this presentation; here we can only present the resulting picture.[1] Figure 1 shows the phase diagram for the two types of dynamics, in the $(\beta J_s, \beta J_\ell)$ plane (note: of the three parameters $\{\beta, J_s, J_\ell\}$ one is redundant). In contrast to models with nearest neighbour interactions only ($J_\ell = 0$, where no pattern recall ever will occur), and to models with mean-field interactions only ($J_s = 0$, where pattern recall can occur), the combination of the two interaction types leads to qualitatively new modes of operation. This especially in the competition region, where $J_\ell > 0$ and $J_s < 0$ (Hebbian long-range synapses, combined with anti-Hebbian short range ones). The novel features of the diagram can play a useful role: phase coexistence ensures that only sufficiently strong recall cues will evoke pattern recognition; the discontinuity of the transition subsequently ensures that in the latter case the recall will be of a substantial quality. In the case of parallel dynamics, similar statements can be made in the opposite region of synaptic competition, but now involving 2-cycles. Since figure 1 cannot show the zero noise region ($\beta = T^{-1} = \infty$), we have also drawn the interesting competition region of the sequential dynamics phase diagram in the $(J_\ell, T)$ plane, for $J_s = -1$ (see figure 3, left picture). At $T = 0$ one finds coexistence of recall states ($m \neq 0$) and non-recall states ($m = 0$) for any $J_\ell > 0$, as soon as $J_s < 0$. In the same figure (right picture) we show the magnitude of the discontinuity in the order parameter $m$ at the discontinuous transition, as a function of $\beta J_\ell$.

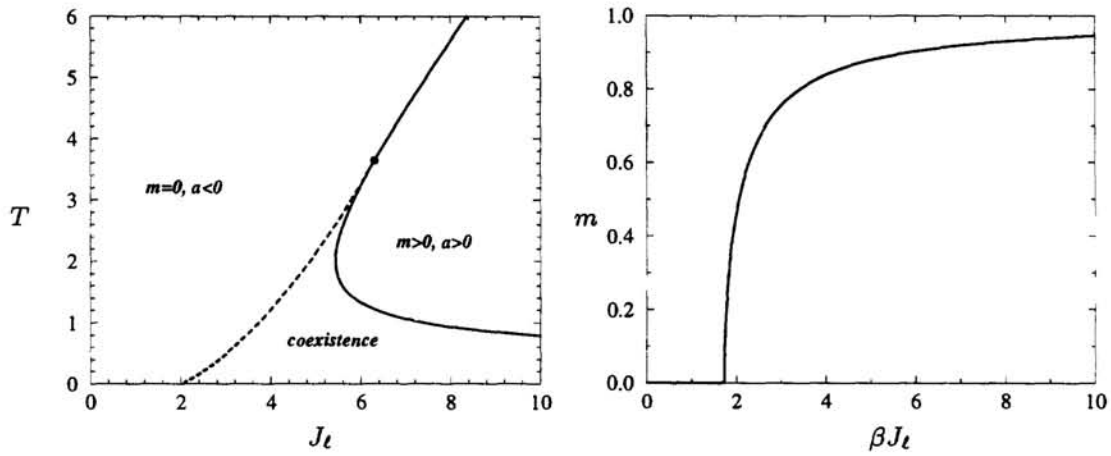

Figure 2: Left picture: alternative presentation of the competition region of the sequential dynamics phase diagram in figure 1. Here the system states and transitions are drawn in the $(J_\ell, T)$ plane $(T = \beta^{-1})$, for $J_s = -1$. Right picture: the magnitude of the 'jump' of the overlap $m$ along the discontinuous transition line, as a function of $\beta J_\ell$.

The fact that for parallel dynamics one finds 2-cycles in the lower left corner of the phase diagram (figure 1) can be inferred from the exact dynamical solution available along the line $J_s = 0$ (see e.g. [4]), provided by the deterministic map $m(t+1) = \tanh[\beta J_\ell m(t)]$.

Finally we show, by way of further illustration of the coexistence mechanism, the value of reduced exponent $\phi_{\text{seq}}(m)$ given in (9), evaluated upon elimination of the auxiliary order parameter $\hat{m}$: $\phi(m) \equiv \phi_{\text{seq}}(m, im J_\ell)$. The result, for the parameter choice $(\beta, J_\ell) = (2, 3)$ and for three different short-range coupling stengths (corresponding to the three phase regimes: non-zero recall, coexistence and zero recall) is given in figure 3. In the same figure we also give the sequential dynamics bifurcation diagram displaying the value(s) of the overlap $m$ as a function of $\beta J_\ell$ and for $\beta J_s = -0.6$ (a line crossing all three phase regimes in figure (1)).

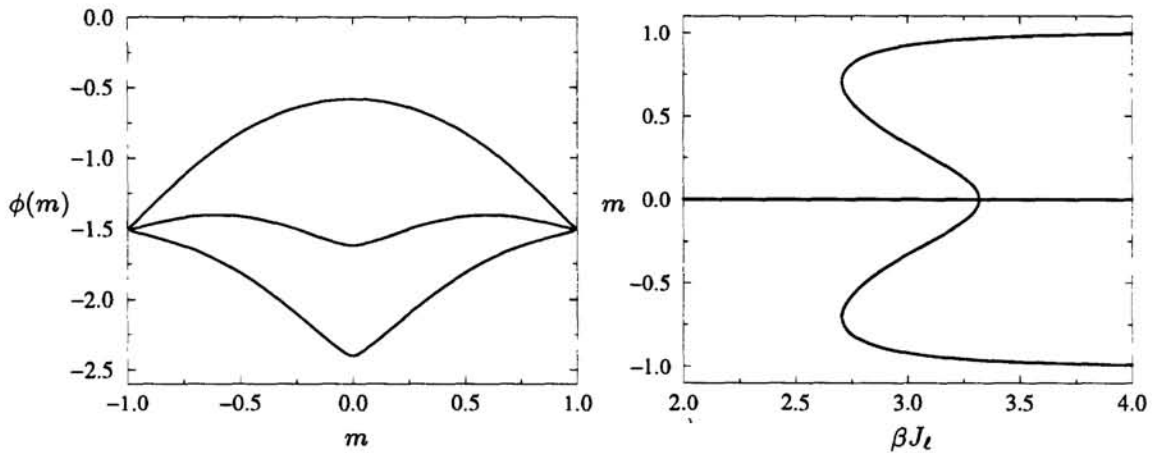

Figure 3: Left: Graph of the reduced exponent $\phi(m) = \phi_{\text{seq}}(m, im J_\ell)$ for the parameter choice $(\beta, J_\ell) = (2, 3)$. The three lines (from upper to lower: $J_s = -1.2, -0.8, -0.2$) correspond to regimes where (i) $m \cdot \neq 0$ only (ii) coexistence of trivial and non-trivial recall states occurs and (iii) $m = 0$ only. Right: Sequential dynamics bifurcation diagram displaying for $\beta J_s = -0.6$ the possible recall solutions. For a critical $\beta J_\ell$ given by (15) $m$ jumps discontinuously to non-zero values. For increasing values of $\beta J_\ell$ the unstable $m \neq 0$ solutions converge towards the trivial one until $\beta J_\ell = \exp(1.2)$ where a continuous phase transition takes place and $m = 0$ becomes unstable.

## 6   DISCUSSION

In this paper we have presented exact analytical equilibrium solutions, for sequential and parallel neuronal dynamics, for a class of recurrent neural network models which allow for a tunable competition between short-range synapses (operating between nearest neighbours only) and long-range ones (operating between any pair of neurons). The present models have been solved exactly by a combination of mean-field techniques and transfer matrix techniques. We found that there exist regions in parameter space where discontinuous transitions take place between states without pattern recall and either states of partial/full pattern recall or 2-cycles. These regions correspond to the ranges of the network parameters where the competition is most evident, for instance, where one has strongly *excitatory* long-range interactions and strongly *inhibitory* short-range ones. In addition this competition is found to generate a coexistence of pattern recall states or 2-cycles with the non-recall state, which (in turn) induces a dependence on initial conditions of whether or not recall will at all take place.

This study is, however, only a first step. In a similar fashion one can now study more complicated systems, where (in addition to the long-range synapses) the short-range synapses reach beyond nearest neighbours, or where the system is effectively on a two-dimensional (rather than one-dimensional) array. Such models can still be solved using the techniques employed here. A different type of generalisation would be to allow for a competition between synapses which would not all be of a Hebbian form, e.g. by having long-range Hebbian synapses (modeling processing via pyramidal neurons) in combination with short-range inhibitory synapses without any effect of learning (modeling processing via simple inhibitory inter-neurons). In addition, one could increase the complexity of the model by storing more than just a single pattern. In the latter types of models the various pattern components can no longer be transformed away, and one has to turn to the methods of random field Ising models (see e.g. [12]).

## Footnotes

[1]Due to the occurrence of imaginary saddle-points in (10) and our strategy to eliminate the variable $\hat{m}$ by using the equation $\partial_m \phi(m, \hat{m}) = 0$, it need not be true that the saddle-point with the lowest value of $\phi(m, \hat{m})$ is the minimum of $\phi$ (complex conjugation can induce curvature sign changes, and in addition the minimum could occur at boundaries or as special limits). Inspection of the status of saddle-points and identification of the physical ones in those cases where there are multiple solutions is thus a somewhat technical issue, details of which will be published elsewhere [11].

## References

[1] D.J. Amit, H. Gutfreund and H. Sompolinsky (1985), *Phys. Rev.* **A32**, 1007-1018

[2] D.J. Amit, H. Gutfreund and H. Sompolinsky (1985), *Phys. Rev. Lett.* **55**, 1530-1533

[3] A.C.C. Coolen and D. Sherrington (1993), in J.G.Taylor (editor) *Mathematical Approaches to Neural Networks*, Elsevier Science Publishers, 293-306

[4] A.C.C. Coolen (1997), *Statistical Mechanics of Neural Networks*, King's College London Lecture Notes

[5] A.C.C. Coolen, S.N. Laughton and D. Sherrington (1996), in D.S. Touretzky, M.C. Mozer and M.E. Hasselmo (eds) *Advances in Neural Information Processing Systems 8*, MIT Press

[6] A. Castellanos, A.C.C. Coolen and L. Viana (1998), *J. Phys. A* **31**, 6615-6634

[7] E. Domany, J.L. van Hemmen and K. Schulten (eds) (1994), *Models of Neural Networks II*, Springer

[8] A.C.C. Coolen and L.G.V.M. Lenders (1992), *J. Phys A* **25**, 2593-2606

[9] J.M. Yeomans (1992), *Statistical Mechanics of Phase Transitions*, Oxford U.P.

[10] P. Peretto (1984), *Biol. Cybern.* **50**, 51-62

[11] N.S. Skantzos and A.C.C. Coolen (1998), *in preparation*

[12] U. Brandt and W. Gross (1978), *Z. Physik B* **31**, 237-245
